# A self-organizing multiple-view representation of 3D objects

**Daphna Weinshall**
Center for Biological
Information Processing
MIT E25-201
Cambridge, MA 02139

**Shimon Edelman**
Center for Biological
Information Processing
MIT E25-201
Cambridge, MA 02139

**Heinrich H. Bülthoff**
Dept. of Cognitive and
Linguistic Sciences
Brown University
Providence, RI 02912

## ABSTRACT

We demonstrate the ability of a two-layer network of thresholded summation units to support representation of 3D objects in which several distinct 2D views are stored for each object. Using unsupervised Hebbian relaxation, the network learned to recognize ten objects from different viewpoints. The training process led to the emergence of compact representations of the specific input views. When tested on novel views of the same objects, the network exhibited a substantial generalization capability. In simulated psychophysical experiments, the network's behavior was qualitatively similar to that of human subjects.

## 1   Background

Model-based object recognition involves, by definition, a comparison between the input image and models of different objects that are internal to the recognition system. The form in which these models are best stored depends on the kind of information available in the input, and on the trade-off between the amount of memory allocated for the storage and the degree of sophistication required of the recognition process.

In computer vision, a distinction can be made between representation schemes that use 3D object-centered coordinate systems and schemes that store viewpoint-specific information such as 2D views of objects. In principle, storing enough 2D views would

allow the system to use simple recognition techniques such as template matching. If only a few views of each object are remembered, the system must have the capability to normalize the appearance of an input object, by carrying out appropriate geometrical transformations, before it can be directly compared to the stored representations.

What representation strategy is employed by the human visual system? The notion that objects are represented in viewpoint-dependent fashion is supported by the finding that commonplace objects are more readily recognized from certain so-called canonical vantage points than from other, random viewpoints (Palmer et al. 1981). Namely, canonical views are identified more quickly (and more accurately) than others, with response times decreasing monotonically with increasing subjective goodness.[1]

The monotonic increase in the recognition latency with misorientation of the object relative to a canonical view prompts the interpretation of the recognition process in terms of a mechanism related to mental rotation. In the classical mental rotation task (see Shepard & Cooper 1982), the subject is required to decide whether two simultaneously presented images are two views of the same 3D object. The average latency of correct response in this task is linearly dependent on the difference in the 3D attitude of the object in the two images. This dependence is commonly accounted for by postulating a process that attempts to rotate the 3D shapes perceived in the two images into congruence before making the identity decision. The rotation process is sometimes claimed to be analog, in the sense that the representation of the object appears to pass through intermediate orientation stages as the rotation progresses (Shepard & Cooper 1982).

Psychological findings seem to support the involvement of some kind of mental rotation in recognition by demonstrating the dependence of recognition latency for an unfamiliar view of an object on the distance to its closest familiar view. There is, however, an important qualification. Practice with specific objects appears to cause this strategy to be abandoned in favor of a more memory-intensive, less time-consuming direct comparison strategy. Under direct comparison, many views of the objects are stored and recognition proceeds in essentially constant time, provided that the presented views are sufficiently close to one of the stored views (Tarr & Pinker 1989, Edelman et al. 1989).

From the preceding outline, it appears that a faithful model of object representation in the human visual system should provide both for the ability to "rotate" 3D objects and for the fast direct-comparison strategy that supersedes mental rotation for highly familiar objects. Surprisingly, it turns out that mental rotation in recognition can be replicated by a self-organizing memory-intensive model based on direct comparison. The rest of the present paper describes such a model, called CLF (conjunctions of localized features; see Edelman & Weinshall 1989).

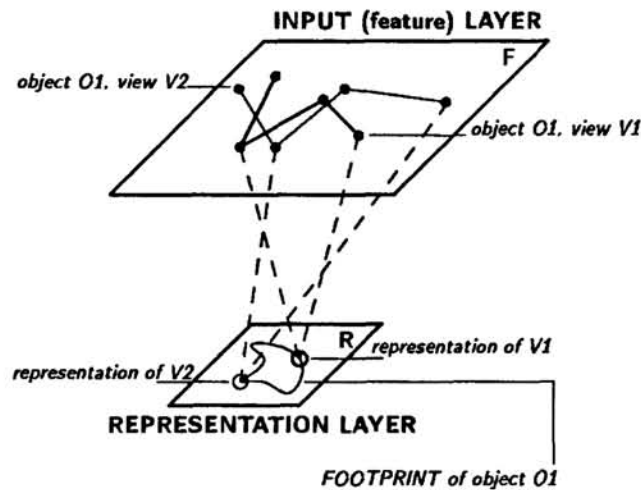

**Figure 1:** The network consists of two layers, **F** (input, or feature, layer) and **R** (representation layer). Only a small part of the projections from **F** to **R** are shown. The network encodes input patterns by making units in the R-layer respond selectively to conjunctions of features localized in the F-layer. The curve connecting the representations of the different views of the same object in R-layer symbolizes the association that builds up between these views as a result of practice.

## 2    The model

The structure of the model appears in Figure 1 (see Edelman & Weinshall 1989 for details). The first (input, or feature) layer of the network is a feature map. In our experiments, vertices of wire-frame objects served as the input features. Every unit in the (feature) F-layer is connected to all units in the second (representation) R-layer. The initial strength of a "vertical" (V) connection between an F-unit and an R-unit decreases monotonically with the "horizontal" distance between the units, according to an inverse square law (which may be considered the first approximation to a Gaussian distribution). In our simulations the size of the F-layer was $64 \times 64$ units and the size of the R-layer – $16 \times 16$ units. Let $(x, y)$ be the coordinates of an F-unit and $(i, j)$ – the coordinates of an R-unit. The initial weight between these two units is $w_{xyij}|_{t=0} = (\sigma[1 + (x - 4i)^2 + (y - 4j)^2])^{-1}$, where $\sigma = 50$ and $(4i, 4j)$ is the point in the F-layer that is directly "above" the R-unit $(i, j)$.

The R-units in the representation layer are connected among themselves by lateral (L) connections, whose initial strength is zero. Whereas the V-connections form the representations of individual views of an object, the L-connections form associations among different views of the same object.

### 2.1    Operation

During training, the input to the model is a sequence of appearances of an object, encoded by the 2D locations of concrete sensory features (vertices) rather than a list

of abstract features. At the first presentation of a stimulus several representation units are active, all with different strengths (due to the initial distribution of vertical connection strengths).

### 2.1.1  Winner Take All

We employ a simple winner-take-all (WTA) mechanism to identify for each view of the input object a few most active R-units, which subsequently are recruited to represent that view. The WTA mechanism works as follows. The net activities of the R-units are uniformly thresholded. Initially, the threshold is high enough to ensure that all activity in the R-layer is suppressed. The threshold is then gradually decreased, by a fixed (multiplicative) amount, until some activity appears in the R-layer. If the decrease rate of the threshold is slow enough, only a few units will remain active at the end of the WTA process. In our implementation, the decrease rate was 0.95. In most cases, only one winner emerged.

Note that although the WTA can be obtained by a simple computation, we prefer the stepwise algorithm above because it has a natural interpretation in biological terms. Such an interpretation requires postulating two mechanisms that operate in parallel. The first mechanism, which looks at the activity of the R-layer, may be thought as a high fan-in OR gate. The second mechanism, which performs uniform adjustable thresholding on all the R-units, is similar to a global bias. Together, they resemble feedback-regulated global arousal networks that are thought to be present, e.g., in the medulla and in the limbic system of the brain (Kandel & Schwartz 1985).[2]

### 2.1.2  Adjustment of weights and thresholds

In the next stage, two changes of weights and thresholds occur that make the currently active R-units (the winners of the WTA stage) selectively responsive to the present view of the input object. First, there is an enhancement of the V-connections from the active (input) F-units to the active R-units (the winners). At the same time, the thresholds of the active R-units are raised, so that at the presentation of a different input these units will be less likely to respond and to be recruited anew. We employ Hebbian relaxation to enhance the V-connections from the input layer to the active R-unit (or units). The connection strength $v_{ab}$ from F-unit $a$ to R-unit $b = (i, j)$ changes by

$$\Delta v_{ab} = \min\left\{\alpha v_{ab} A_a \cdot A_{ij}, v^{max} - v_{ab}\right\} \cdot \frac{v^{max} - v_{ab}}{v^{max}} \tag{1}$$

where $A_{ij}$ is the activation of the R-unit $(i, j)$ after WTA, $v^{max}$ is an upper bound on a connection strength and $\alpha$ is a parameter controlling the rate of convergence. The threshold of a winner R-unit is increased by

$$\Delta T_b = \delta \sum_a \Delta v_{ab} A_a \tag{2}$$

where $\delta \leq 1$. This rule keeps the thresholded activity level of the unit growing while the unit becomes more input specific. As a result, the unit encodes the spatial structure of a specific view, responding selectively to that view after only a few (two or three) presentations.

### 2.1.3    Between-views association

The principle by which specific views of the same object are grouped is that of temporal association. New views of the object appear in a natural order, corresponding to their succession during an arbitrary rotation of the object. The lateral (L) connections in the representation layer are modified by a time-delay Hebbian relaxation. L-connection $w_{bc}$ between R-units $b = (i, j)$ and $c = (l, m)$ that represent successive views is enhanced in proportion to the closeness of their peak activations in time, up to a certain time difference $K$:

$$\Delta w_{bc} = \sum_{|k| < K} AM(b, c) \cdot \gamma_k A_{ij}^t \cdot A_{lm}^{t+k} \cdot \frac{w^{max} - w_{bc}}{w^{max}} \tag{3}$$

The strength of the association between two views is made proportional to a coefficient, $AM(b, c)$, that measures the strength of the apparent motion effect that would ensue if the two views were presented in succession to a human subject (see Edelman & Weinshall 1989).

### 2.1.4    Multiple-view representation

The appearance of a new object is explicitly signalled to the network, so that two different objects do not become associated by this mechanism. The parameter $\gamma_k$ decreases with $|k|$ so that the association is stronger for units whose activation is closer in time. In this manner, a *footprint* of temporally associated view-specific representations is formed in the second layer for each object. Together, the view-specific representations form a distributed multiple-view representation of the object.

## 3    Testing the model

We have subjected the CLF network to simulated experiments, modeled after the experiments of (Edelman et al. 1989). Some of the results of the real and simulated experiments appear in Figures 2 and 3. In the experiments, each of ten novel 3D wire-frame objects served in turn as target. The task was to distinguish between the target and the other nine, non-target, objects. The network was first trained on a set of projections of the target's vertices from 16 evenly spaced viewpoints. After learning the target using Hebbian relaxation as described above, the network

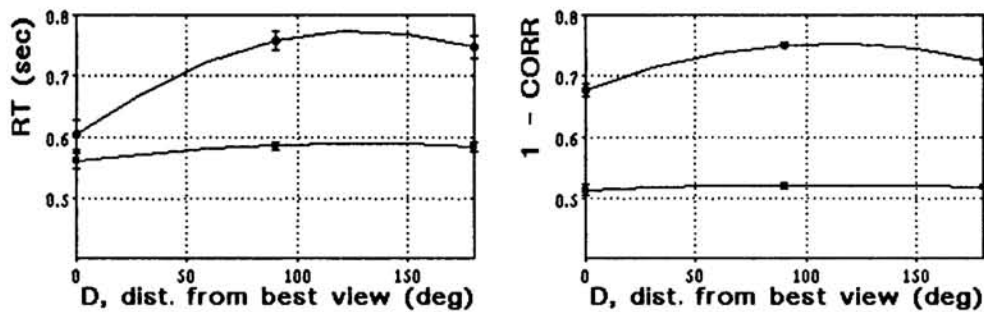

**Figure 3:** Another comparison of human performance (left panel) with that of the CLF model (right panel). Define the **best view** for each object as the view with the shortest RT (highest CORR). If recognition involves rotation to the best (canonical) view, RT or CORR should depend monotonically on $D = D(target, view)$, the distance between the best view and the actually shown view. (The decrease in RT or CORR at $D = 180°$ is due to the fact that for the wire-frame objects used in the experiments the view diametrically opposite the best one is also easily recognized.) For both human subjects and the model, the dependence is clear for the first session of the experiment (upper curves), but disappears with practice (second session – lower curves).

We note that blurring the input prior to its application to the F-layer can significantly extend the generalization ability of the CLF model. Performing autoassociation on a dot pattern blurred with a Gaussian is computationally equivalent to correlating the input with a set of templates, realized as Gaussian receptive fields. This, in turn, appears to be related to interpolation with Radial Basis Functions (Moody & Darken 1989, Poggio & Girosi 1989, Poggio & Edelman 1989).

## 4  Summary

We have described a two-layer network of thresholded summation units which is capable of developing multiple-view representations of 3D objects in an unsupervised fashion, using fast Hebbian learning. Using this network to model the performance of human subjects on similar stimuli, we replicated psychophysical experiments that investigated the phenomena of canonical views and mental rotation. The model's performance closely parallels that of the human subjects, even though the network has no a priori mechanism for "rotating" object representations. In the model, a semblance of rotation is created by progressive activation of object footprints (chains of representation units created through association during training). Practice causes the footprints to lose their linear structure through the creation of secondary association links between random representation units, leading to the disappearance of orientation effects. Our results may indicate that a different interpretation of findings that are usually taken to signify mental rotation is possible. The foot-

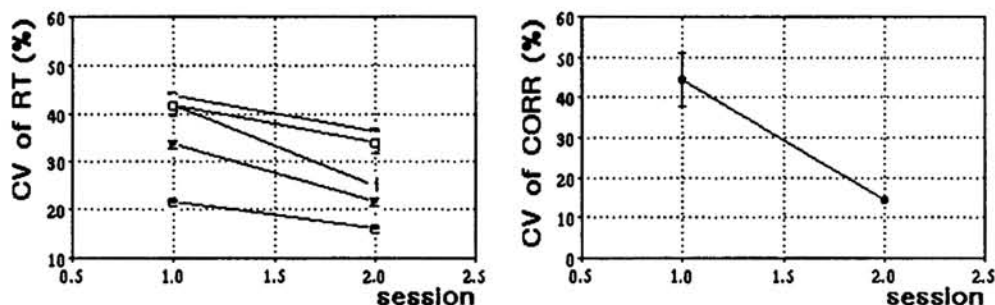

**Figure 2:** Performance of five human subjects (left panel) and of the CLF model (right panel). The variation of the performance measure (for human subjects, response time RT; for the model, correlation CORR between the input and a stored representation) over different views of an object serves as an estimate of the strength of the canonical views phenomenon. In both human subjects and the model, practice appears to reduce the strength of this phenomenon.

was tested on a sequence of inputs, half of which consisted of familiar views of the target, and half of views of other, not necessarily familiar, objects.

The presentation of an input to the F-layer activated units in the representation layer. The activation then spread to other R-units via the L-connections. After a fixed number of lateral activation cycles, we correlated the resulting pattern of activity with footprints of objects learned so far. The object whose footprint yielded the highest correlation was recognized by definition. In the beginning of the testing stage, this correlation, which served as an analog of response time,[3] exhibited strong dependence on object orientation, replicating the effect of mental rotation in recognition. During testing, successive activation of R-units through association strengthened the L-connection between them, leading to an obliteration of the linear structure of R-unit sequences responsible for mental rotation effects.

### 3.1   Generalization to novel views

The usefulness of a recognition scheme based on multiple-view representation depends on its ability to classify correctly novel views of familiar objects. To assess the generalization ability of the CLF network, we have tested it on views obtained by rotating the objects away from learned views by as much as 23° (see Figure 4). The classification rate was better than chance for the entire range of rotation. For rotations of up to 4° it was close to perfect, decreasing to 30% at 23° (chance level was 10% because we have used ten objects). One may compare this result with the finding (Rock & DiVita 1987) that people have difficulties in recognizing or imagining wire-frame objects in a novel orientation that differs by more than 30° from a familiar one.

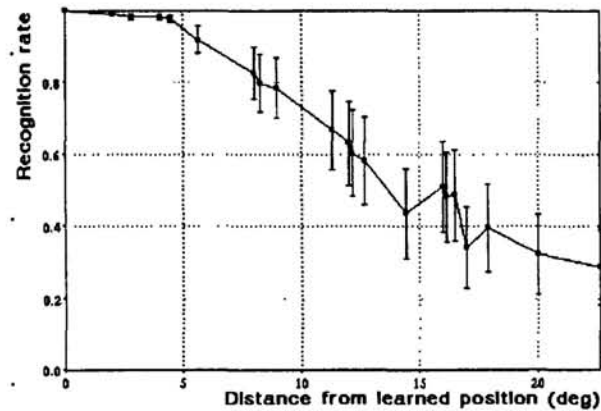

**Figure 4:** Performance of the network on novel orientations of familiar objects (mean of 10 objects, bars denote the variance).

prints formed in the representation layer in our model provide a hint as to what the substrate upon which the mental rotation phenomena are based may look like.

## Footnotes

[1] Canonical views of objects can be reliably identified in subjective judgement as well as in recognition tasks. For example, when asked to form a mental image of an object, people usually imagine it as seen from a canonical perspective.

[2] The relationship of this approach to other WTA algorithms is discussed in Edelman & Weinshall 1989.

[3] The justification for this use of correlation appears in Edelman & Weinshall 1989.

# References

[1] S. Edelman, H. Bülthoff, and D. Weinshall. Stimulus familiarity determines recognition strategy for novel 3D objects. MIT A.I. Memo No. 1138, 1989.

[2] S. Edelman and D. Weinshall. A self-organizing multiple-view representation of 3D objects. MIT A.I. Memo No. 1146, 1989.

[3] E. R. Kandel and J. H. Schwartz. *Principles of neural science*. Elsevier, 1985.

[4] J. Moody and C. Darken. Fast learning in networks of locally tuned processing units. *Neural Computation*, 1:281–289, 1989.

[5] S. Palmer, E. Rosch, and P. Chase. Canonical perspective and the perception of objects. In J. Long and A. Baddeley, eds, *Attn. & Perf. IX*, 135–151. Erlbaum, 1981.

[6] T. Poggio and S. Edelman. A network that learns to recognize 3D objects. *Nature*, 1989, in press.

[7] T. Poggio and F. Girosi. A theory of networks for approximation and learning. MIT A.I. Memo No. 1140, 1989.

[8] I. Rock and J. DiVita. A case of viewer-centered object perception. *Cognitive Psychology*, 19:280–293, 1987.

[9] R. N. Shepard and L. A. Cooper. *Mental images and their transformations*. MIT Press, 1982.

[10] M. Tarr and S. Pinker. Mental rotation and orientation-dependence in shape recognition. *Cognitive Psychology*, 21, 1989.
